# Reorganisation of Somatosensory Cortex after Tactile Training

Rasmus S. Petersen        John G. Taylor

Centre for Neural Networks, King's College London
Strand, London WC2R 2LS, UK

## Abstract

Topographic maps in primary areas of mammalian cerebral cortex reorganise as a result of behavioural training. The nature of this reorganisation seems consistent with the behaviour of competitive neural networks, as has been demonstrated in the past by computer simulation. We model tactile training on the hand representation in primate somatosensory cortex, using the Neural Field Theory of Amari and his colleagues. Expressions for changes in both receptive field size and magnification factor are derived, which are consistent with owl monkey experiments and make a prediction which goes beyond them.

## 1. INTRODUCTION

The primary cortical areas of mammals are now known to be plastic throughout life; reviewed recently by Kaas(1995). The problem of how and why the underlying learning processes work is an exciting one, for which neural network modelling appears well suited. In this contribution, we model the long-term effects of tactile training (Jenkins et al, 1990) on the functional organisation of monkey primary somatosensory cortex, by perturbing a topographic net (Takeuchi and Amari, 1979).

### 1.1 ADAPTATION IN ADULT SOMATOSENSORY CORTEX

Light touch activates skin receptors which in primates are mapped, largely topographically, in area 3b. In a series of papers, Merzenich and colleagues describe how area 3b becomes reorganised following peripheral nerve damage (Merzenich et al, 1983a; 1983b) or digit amputation (Merzenich et al, 1984). The underlying learning processes may also explain the phenomenon of phantom limb "telescoping" (Haber, 1955). Recent advances in brain scanning are beginning to make them observable even in the human brain (Mogilner et al, 1993).

### 1.2 ADAPTATION ASSOCIATED WITH TACTILE TRAINING

Jenkins et al trained owl monkeys to maintain contact with a rotating disk. The apparatus was arranged so that success eventually involved touching the disk with only the digit tips. Hence these regions received selective stimulation. Some time after training had been completed electro-physiological recordings were made from area 3b. These revealed an increase in Magnification Factor (MF) for the stimulated skin and a decrease in

the size of Receptive Fields (RFs) for that region. The net territory gained for light touch of the digit tips came from area 3a and/or the face region of area 3b, but details of any changes in these representations were not reported.

## 2. THEORETICAL FRAMEWORK

### 2.1 PREVIOUS WORK

Takeuchi and Amari(1979), Ritter and Schulten(1986), Pearson et al(1987) and Grajski and Merzenich(1990) have all modelled amputation/denervation by computer simulation of competitive neural networks with various Hebbian weight dynamics. Grajski and Merzenich(1990) also modelled the data of Jenkins et al. We build on this research within the *Neural Field Theory* framework (Amari, 1977; Takeuchi and Amari, 1979; Amari, 1980) of the *Neural Activity Model* of Willshaw and von der Malsburg(1976).

### 2.2 NEURAL ACTIVITY MODEL

Consider a "cortical" network of simple, laterally connected neurons. Neurons sum inputs linearly and output a sigmoidal function of this sum. The lateral connections are excitatory at short distances and inhibitory at longer ones. Such a network is competitive: the steady state consists of blobs of activity centred around those neurons locally receiving the greatest afferent input (Amari, 1977). The range of the competition is limited by the range of the lateral inhibition.

Suppose now that the afferent synapses adapt in a Hebbian manner to stimuli that are localised in the sensory array; the lateral ones are fixed. Willshaw and von der Malsburg(1976) showed by computer simulation that this network is able to form a topographic map of the sensory array. Takeuchi and Amari(1979) amended the Willshaw-Malsburg model slightly: neurons possess an *adaptive firing threshold* in order to prevent synaptic weight explosion, rather than the more usual mechanism of weight normalisation. They proved that a topographic mapping is stable under certain conditions.

### 2.3 TAKEUCHI-AMARI THEORY

Consider a one-dimensional model. The membrane dynamics are:

$$\frac{\partial u(x,y,t)}{\partial t} = -u(x,y,t) + \int s(x,y',t)a(y-y')dy' - $$

$$s_0(x,t)a_0 + \int w(x-x')f[u(x',y,t)]dx' - h \qquad (1)$$

Here $u(x,y,t)$ is the membrane potential at time $t$ for point $x$ when a stimulus centred at $y$ is being presented; $h$ is a positive resting potential; $w(z)$ is the lateral inhibitory weight between two points in the neural field separated by a distance $z$ - positive for small $|z|$ and negative for larger $|z|$; $s(x,y,t)$ is the excitatory synaptic weight from $y$ to $x$ at time $t$ and $s_0(x,t)$ is an inhibitory weight from a tonically active inhibitory input $a_0$ to $x$ at time $t$ - it is the adaptive firing threshold. $f[u]$ is a binary threshold function that maps positive membrane potentials to 1 and non-positive ones to 0.

Idealised, point-like stimuli are assumed, which "spread out" somewhat on the sensory surface or subcortically. The spreading process is assumed to be independent of $y$ and is described in the same coordinates. It is represented by the function $a(y-y')$, which describes the effect of a point input at $y$ spreading to the point $y'$. This is a decreasing, positive, symmetric function of $|y-y'|$. With this type of input, the steady-state activity of the network is a single blob, localised around the neuron with maximum afferent input.

The afferent synaptic weights adapt in a leaky Hebbian manner but with a time constant much larger than that of the membrane dynamics (1). Effectively this means that learning occurs on the steady state of the membrane dynamics. The following averaged weight dynamics can be justified (Takeuchi and Amari, 1979; Geman 1979):

$$\frac{\partial s(x,y,t)}{\partial t} = -s(x,y,t) + b \int p(y')a(y-y')\,\mathrm{f}[\hat{u}(x,y')]dy'$$

$$\frac{\partial s_0(x,y,t)}{\partial t} = -s_0(x,y,t) + b'\,a_0 \int p(y')\,\mathrm{f}[\hat{u}(x,y')]dy' \tag{2}$$

where $\hat{u}(x,y')$ is the steady-state of the membrane dynamics at $x$ given a stimulus at $y'$ and $p(y')$ is the probability of a stimulus at $y'$; $b$, $b'$ are constants.

Empirically, the "classical" Receptive Field (RF) of a neuron is defined as the region of the input field within which localised stimulation causes change in its activity. This concept can be modelled in neural field theory as: the RF of a neuron at $x$ is the portion of the input field within which a stimulus evokes a positive membrane potential (inhibitory RFs are not considered). If the neural field is a continuous map of the sensory surface then the RF of a neuron is fully described by its two borders $r_1(x)$, $r_2(x)$, defined formally:

$$\hat{u}\big(x, r_i(x)\big) = 0 \qquad i = 1,2 \tag{3}$$

which are illustrated in figure 1.

Let RF size and RF position be denoted respectively by the functions $r(x)$ and $m(x)$, which represent experimentally measurable quantities. In terms of the border functions they can be expressed:

$$r(x) = r_2(x) - r_1(x)$$

$$m(x) = \tfrac{1}{2}\big(r_1(x) + r_2(x)\big) \tag{4}$$

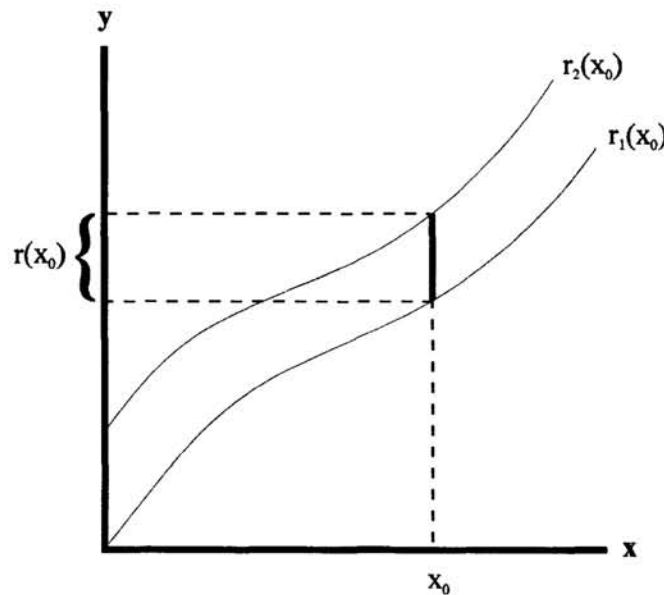

Figure 1: RF boundaries as a function of position in the neural field, for a topographically ordered network. Only the region in-between $r_1(x)$ and $r_2(x)$ has positive steady-state membrane potential $\hat{u}(x,y)$. $r_1(x)$ and $r_2(x)$ are defined by the condition $\hat{u}(x,r_i(x))=0$ for $i=1,2$.

Using (1), (2) and the definition (3), Takeuchi and Amari(1979) derived dynamical equations for the change in RF borders due to learning. In the case of uniform stimulus probability, they found solutions for the steady-state RF border functions. With periodic boundary conditions, the basic solution is a linear map with constant RF size:

$$r(x) = r_0 = \text{const} \qquad r_1^{uni}(x) = \rho x$$
$$m(x) = \rho x + \tfrac{1}{2} r_0 \qquad r_2^{uni}(x) = \rho x + r_0 \tag{5}$$

This means that both RF size and activity blob size are uniform across the network and that RF position $m(x)$ is a linear function of network location. (The value of $\rho$ is determined by boundary conditions; $r_0$ is then determined from the joint equilibrium of (1), (2)). The inverse of the RF position function, denoted by $m^{-1}(y)$, is the centre of the cortical active region caused by a stimulus centred at $y$. The change in $m^{-1}(y)$ over a unit interval in the input field is, by empirical definition, the cortical magnification factor (MF). Here we model MF as the rate of change of $m^{-1}(y)$. The MF for the system described by (5) is:

$$\frac{d}{dy} m^{-1}(y) = \rho^{-1} \tag{6}$$

## 3. ANALYSIS OF TACTILE TRAINING

### 3.1 TRAINING MODEL AND ASSUMPTIONS

Jenkins et al's training sessions caused an increase in the relative frequency of stimulation to the finger tips, and hence a decrease in relative frequency of stimulation elsewhere. Over a long time, we can express this fact as a localised change in stimulus probability (figure 2). (This is not sufficient to cause cortical reorganisation - Recanzone et al(1992) showed that attention to the stimulation is vital. We consider only attended stimulation in this model). To account for such data it is clearly necessary to analyse *non-uniform stimulus probabilities*, which demands extending the results of Takeuchi and Amari. Unfortunately, it seems to be hard to obtain general results. However, a perturbation analysis around the uniform probability solution (5) is possible.

To proceed in this way, we must be able to assume that the change in the stimulus probability density function away from uniformity is small. This reasoning is expressed by the following equation:

$$p(y) = p_0 + \varepsilon \tilde{p}(y) \tag{7}$$

where $p(y)$ is the new stimulus probability in terms of the uniform one and a perturbation due to training: $\varepsilon$ is a small constant. The effect of the perturbation is to ease the weight dynamics (2) away from the solution (5) to a new steady-state. Our goal is to discover the effect of this on the RF border functions, and hence for RF size and MF.

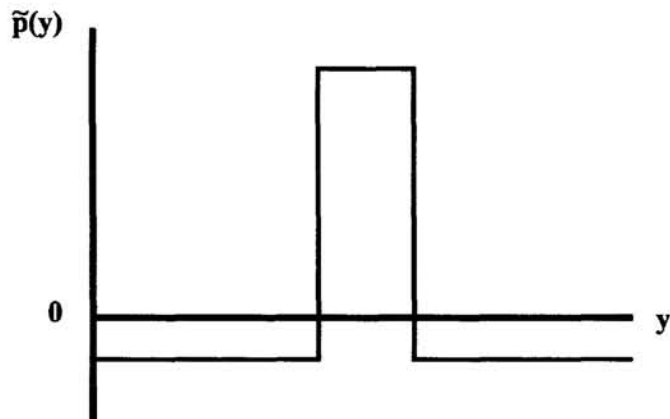

Figure 2: The type of change in stimulus probability density that we assume to model the effects of behavioural training.

## 3.2 PERTURBATION ANALYSIS

### 3.2.1 General Case

For a small enough perturbation, the effect on the RF borders and on the activity blob size ought also to be small. We consider effects to first order in $\varepsilon$, seeking new solutions of the form:

$$r_i^{per}(x) = r_i^{uni}(x) + \varepsilon \tilde{r}_i(x) \qquad i = 1,2 \qquad \begin{aligned} \tilde{r}(x) &= \tilde{r}_2(x) - \tilde{r}_1(x) \\ \tilde{m}(x) &= \tfrac{1}{2}\left(\tilde{r}_1(x) + \tilde{r}_2(x)\right) \end{aligned} \tag{8}$$

where the superscript *per* denotes the new, perturbed equilibrium and *uni* denotes the unperturbed, uniform probability equilibrium. Using (1) and (2) in (3) for the post-training RF borders, expanding to first order in $\varepsilon$, a pair of difference equations may be obtained for the changes in RF borders. It is convenient to define the following terms:

$$A_1(x) = \int_0^{r_0} \tilde{p}(y + \rho x)k(y)dy - b' a_0^2 \int_{r_1^{uni}(x)}^{r_2^{uni}(x)} \tilde{p}(y)dy$$

$$A_2(x) = \int_{-r_0}^{0} \tilde{p}(y + \rho x + r_0)k(y)dy - b' a_0^2 \int_{r_1^{uni}(x)}^{r_2^{uni}(x)} \tilde{p}(y)dy$$

$$k(y) = b\int a(y - y')a(y')dy' \tag{9}$$

$$B = b' a_0^2 p_0 - k(r_0)p_0 > 0$$

$$C = w\left(\rho^{-1}r_0\right)\rho^{-1} < 0$$

where the signs of $B$ and $C$ arise due to stability conditions (Amari, 1977; Takeuchi and Amari, 1979). In terms of RF size and RF position (4), the general result is:

$$B\Delta^2 \tilde{r}(x) = \Delta(\Delta + 1)A_1(x) - \Delta A_2(x)$$

$$BC\Delta^2 \tilde{m}(x) = \left(B - C - \tfrac{1}{2}C\Delta\right)(\Delta + 1)A_1(x) + \left(C - B + \tfrac{1}{2}(C - 2B)\Delta\right)A_2(x) \tag{10}$$

where $\Delta$ is the difference operator:

$$\Delta \, \mathrm{f}(x) = \mathrm{f}\left(x + \rho^{-1}r_0\right) - \mathrm{f}(x) \tag{11}$$

### 3.2.2 Particular Case

The second order difference equations (10) are rather opaque. This is partly due to coupling in $y$ caused by the auto-correlation function $k(y)$: (10) simplifies considerably if very narrow stimuli are assumed - $a(y) = \delta(y)$ (see also Amari, 1980). For periodic boundary conditions:

$$\frac{\tilde{r}(x)}{r_0} = -\frac{1}{p_0 r_0} \int_{r_1^{uni}(x)}^{r_2^{uni}(x)} \tilde{p}(y)dy$$

$$\frac{d}{dy}\tilde{m}^{-1}(y) \approx \frac{1}{2p_0 \rho r_0} \int_{y-r_0}^{y+r_0} \tilde{p}(y')dy' - \frac{ba(0)^2}{w\left(\rho^{-1}r_0\right)r_0}\tilde{p}(y) \tag{12}$$

where:

$$m^{-1\,post}(y) = m^{-1\,pre}(y) + \varepsilon \tilde{m}^{-1}(y)$$
$$= \rho^{-1}\left(y - \tfrac{1}{2} r_0\right) + \varepsilon \tilde{m}^{-1}(y) \tag{13}$$

and we have used the crude approximation:

$$\frac{d}{dx}\tilde{m}(x) \approx \frac{1}{l_0}\Delta m\left(x - \tfrac{1}{2}\rho^{-1}r_0\right) \tag{14}$$

which demands smoothness on the scale of $l_0$. However, for perturbations like that sketched in figure 2, this is sufficient to tell us about the constant regions of MF. (We would not expect to be able to model the data in the transition region in any case, as its form is too dependent upon fine detail of the model).

Our results (12) show that the change in RF size of a neuron is simply minus the total change in stimulus probability over its RF. Hence RF size decreases where p(y) increases and *vice versa*. Conversely, the change in MF at a given stimulus location is roughly the local average change in stimulus probability there. Note that changes in RF size correlate inversely with changes in MF. Figure 3 is a sketch of these results for the perturbation of figure 2.

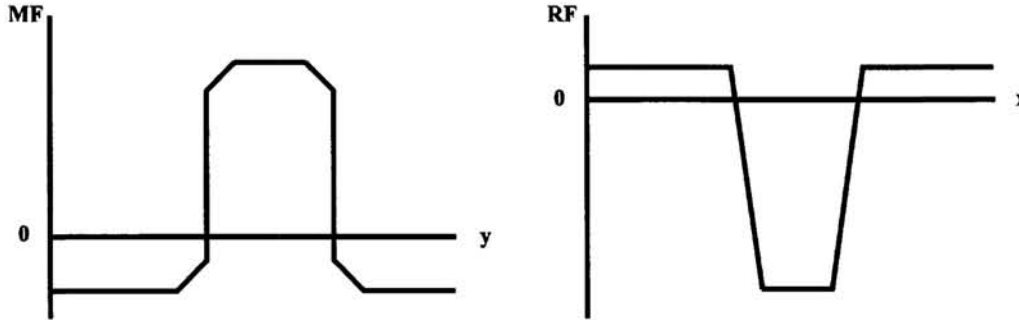

Figure 3: Results of perturbation analysis for how behavioural training (figure 2) changes RF size and MF respectively, in the case where stimulus width can be neglected. For MF - due to the approximation (14) - predictions do not apply near the transitions.

## 4. DISCUSSION

Equations (12) are the results of our model for RF size and MF after area 3b has fully adapted to the behavioural task, in the case where stimulus width can be neglected. They appear to be fully consistent with the data of Jenkins et al described above: RF size decreases in the region of cortex selective for the stimulated body part and the MF for this body part increases. Our analysis also makes a specific prediction that goes beyond Jenkins et al's data, directly due to the inverse relationship between changes in RF size and those in MF. Within the regions that *surrender* territory to the entrained finger tips (sometimes the face region), for which MF decreases, RF sizes should *increase*.

Surprisingly perhaps, these changes in RF size are *not* due to adaptation of the afferent weights $s(x,y)$. The changes are rather due to the *adaptive threshold* term $s_0(x)$. This point will be discussed more fully elsewhere.

A limitation of our analysis is the assumption that the change in stimulus probability is in some sense small. Such an approximation may be reasonable for behavioural training but seems less so as regards important experimental protocols like amputation or denervation. Evidently a more general analysis would be highly desirable.

## 5. CONCLUSION

We have analysed a system with three interacting features: lateral inhibitory interactions; Hebbian adaptivity of afferent synapses and an adaptive firing threshold. Our results indicate that such a system can account for the data of Jenkins et al, concerning the response of adult somatosensory cortex to the changing environmental demands imposed by tactile training. The analysis also brings out a prediction of the model, that may be testable.

### Acknowledgements

RSP is very grateful for a travel stipend from the NIPS Foundation and for a Nick Hughes bursary from the School of Physical Sciences and Engineering, King's College London, that enabled him to participate in the conference.

### References

Amari S. (1977) *Biol. Cybern.* **27** 77-87

Amari S. (1980) *Bull. Math. Biology* **42** 339-364

Geman S. (1979) *SIAM J. App. Math.* **36** 86-105

Grajski K.A., Merzenich M.M. (1990) in *Neural Information Processing Systems* **2** Touretzky D.S. (Ed) 52-59

Haber W.B. (1955) *J. Psychol.* **40** 115-123

Jenkins W.M., Merzenich M.M., Ochs M.T., Allard T., Guíc-Robles E. (1990) *J. Neurophysiol.* **63** 82-104

Kaas J.H. (1995) in *The Cognitive Neurosciences* Gazzaniga M.S. (Ed ic) 51-71

Merzenich M.M., Kaas J.H., Wall J.T., Nelson R.J., Sur M., Felleman D.J. (1983a) *Neuroscience* **8** 35-55

Merzenich M.M., Kaas J.H., Wall J.T., Sur M., Nelson R.J., Felleman D.J. (1983b) *Neuroscience* **10** 639-665

Merzenich M.M., Nelson R.J., Stryker M.P., Cynader M.S., Schoppmann A., Zook J.M. (1984) *J. Comp. Neurol.* **224** 591-605

Mogilner A., Grossman A.T., Ribrary V., Joliot M., Volmann J., Rapaport D., Beasley R., Llinás R. (1993) *Proc. Natl. Acad. Sci. USA* **90** 3593-3597

Pearson J.C., Finkel L.H., Edelman G.M. (1987) *J. Neurosci.* **12** 4209-4223

Recanzone G.H., Merzenich M.M., Jenkins W.M., Grajski K.A., Dinse H.R. (1992) *J. Neurophysiol.* **67** 1031-1056

Ritter H., Schulten K. (1986) *Biol. Cybern.* **54** 99-106

Takeuchi A., Amari S. (1979) *Biol. Cybern.* **35** 63-72

Willshaw D.J., von der Malsburg C. (1976) *Proc. R. Soc. Lond.* **B194** 203-243